# HOW THE CATFISH TRACKS ITS PREY: AN INTERACTIVE "PIPELINED" PROCESSING SYSTEM MAY DIRECT FORAGING VIA RETICULOSPINAL NEURONS.

Jagmeet S. Kanwal
Dept. of Cellular & Structural Biology, Univ. of Colorado, Sch. of Medicine, 4200 East, Ninth Ave., Denver, CO 80262.

## ABSTRACT

Ictalurid catfish use a highly developed gustatory system to localize, track and acquire food from their aquatic environment. The neural organization of the gustatory system illustrates well the importance of the four fundamental ingredients (representation, architecture, search and knowledge) of an "intelligent" system. In addition, the "pipelined" design of architecture illustrates how a goal-directed system effectively utilizes interactive feedback from its environment. Anatomical analysis of neural networks involved in target-tracking indicated that reticular neurons within the medullary region of the brainstem, mediate connections between the gustatory (sensory) inputs and the motor outputs of the spinal cord. Electrophysiological analysis suggested that these neurons integrate selective spatio-temporal patterns of sensory input transduced through a rapidly adapting-type peripheral filter (responding tonically only to a continuously increasing stimulus concentration). The connectivity and response patterns of reticular cells and the nature of the peripheral taste response suggest a unique "gustation-seeking" function of reticulospinal cells, which may enable a catfish to continuously track a stimulus source once its directionality has been computed.

## INTRODUCTION

Food search is an example of a broad class of behaviors generally classified as goal-directed behaviors. Goal-directed behavior is frequently exhibited by animals, humans and some machines. Although a preprogrammed, hard-wired machine may achieve a particular goal in a relatively short time, the general and heuristic nature of complex goal-directed tasks, however, is best exhibited by animals and best studied in some of the less advanced animal species, such as fishes, where anatomical, electro-physiological and behavioral analyses can be performed relatively accurately and easily.

Food search, which may lead to food acquisition and ingestion, is critical for the survival of an organism and, therefore, only highly successful systems are selected during the evolution of a species. The act of food search may be classified into two distinct phases, (i) orientation, and (ii) tracking (navigation and homing). In the channel catfish (the animal model utilized for this study), locomotion (swimming) is primarily controlled by the large forked caudal fin, which also mediates turning and directional swimming.

Both these forms of movement, which constitute the essential movements of target-tracking, involve control of the hypaxial/epiaxial muscles of the flank. The alternate contraction of these muscles causes caudal fin undulations. Each cycle of the caudal fin undulation provides either a symmetrical or an asymmetrical bilateral thrust. The former provides a net thrust forward, along the longitudinal axis of the fish causing it to move ahead, while the latter biases the direction of movement towards the right or left side of the fish.

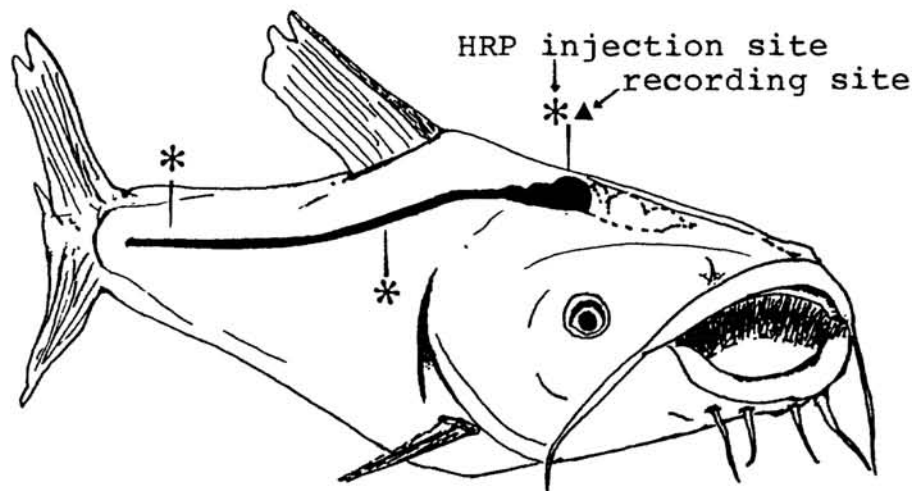

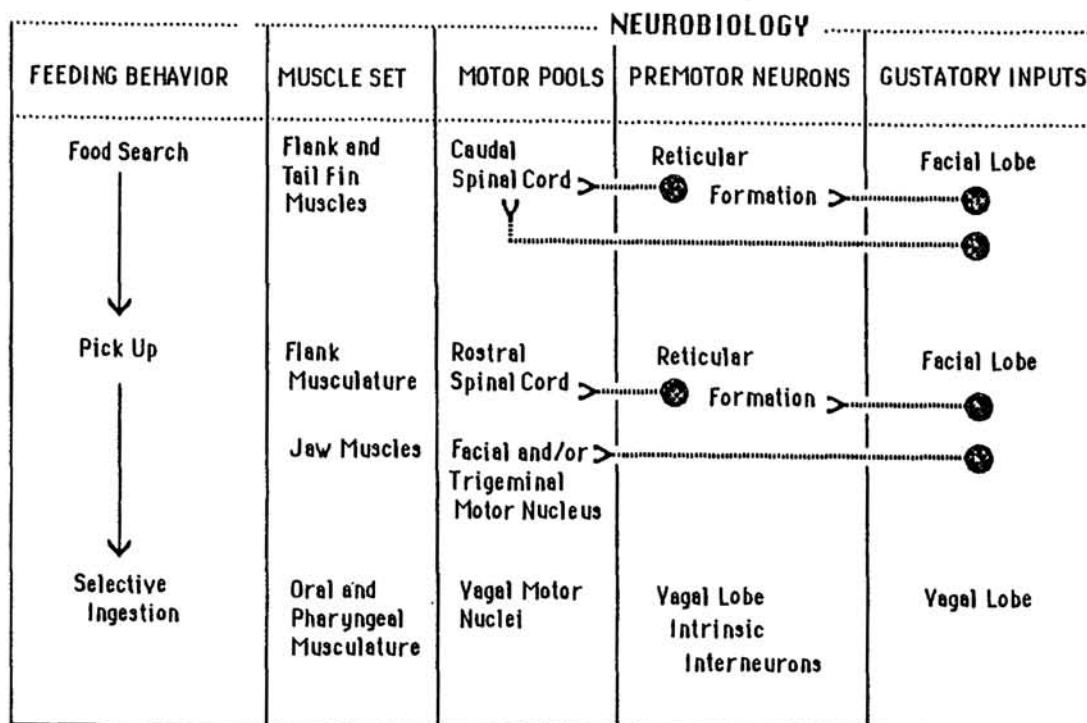

| FEEDING BEHAVIOR | MUSCLE SET | NEUROBIOLOGY | | |
|---|---|---|---|---|
| | | MOTOR POOLS | PREMOTOR NEURONS | GUSTATORY INPUTS |
| Food Search | Flank and Tail Fin Muscles | Caudal Spinal Cord | Reticular Formation | Facial Lobe |
| Pick Up | Flank Musculature | Rostral Spinal Cord | Reticular Formation | Facial Lobe |
| | Jaw Muscles | Facial and/or Trigeminal Motor Nucleus | | |
| Selective Ingestion | Oral and Pharyngeal Musculature | Vagal Motor Nuclei | Vagal Lobe Intrinsic Interneurons | Vagal Lobe |

Fig. 1. Schematic representation of possible pathways for the gustatory modulation of foraging in the catfish.

Ictalurid catfishes possess a well developed gustatory system and use it to locate and acquire food from their aquatic environment[1,2,3]. Behavioral evidence also indicates that ictalurid catfishes can detect small intensity (stimulus concentration) differences across their barbels (interbarbel intensity differences), and may use this or other extraoral taste information to compute the directionality in space and track a gustatory stimulus source [1]. In other words, based upon the analysis of locomotion, it may be inferred that during food search, the gustatory sense of the catfish influences the duration and degree of asymmetrical or symmetrical undulations of the caudal fin, besides controlling reflex turns of the head and flank. Since directional swimming is ultimately dependent upon movement of the large caudal fin it may be postulated that, if the gustatory system is to coordinate food tracking, gustato-spinal connections exist upto the level of the caudal fin of the catfish (fig. 1).

The objectives of this study were (i) to reconsider the functional organization of the gustatory system within the costraints of the four fundamental ingredients (representation, architecture, search and knowledge) of a naturally or artificially "intelligent" agent, (ii) to test the existence of the postulated gustato-spinal connections, and (iii) to delineate as far as possible, using neuroanatomical and electrophysiological techniques, the neural mechanism/s involved in the control of goal-directed (foraging) behavior.

## ORGANIZATIONAL CONSIDERATIONS

I. REPRESENTATION

Representation refers to the translation of a particular task into information structures and information processes and determines to a great extent the efficiency and efficacy with which a solution to the task can be generated[4]. The elaborate and highly sensitive taste system of an ictalurid catfish consists of an extensive array of chemo- and mechanosensory receptors distributed over most of the extraoral as well as oral regions of the epithelium[2,5]. Peripherally, branches of the facial nerve (which innervates all extraoral taste buds) respond to a wide range of stimulus (amino acids) concentrations[6,7,8] i.e. from $10^{-9}$M to $10^{-3}$M. The taste activity however, adapts rapidly (phasic response) to ongoing stimulation of the same concentration (Fig. 2) and responds tonically only to continuously increasing concentrations of stimuli, such as L-arginine and L-alanine.

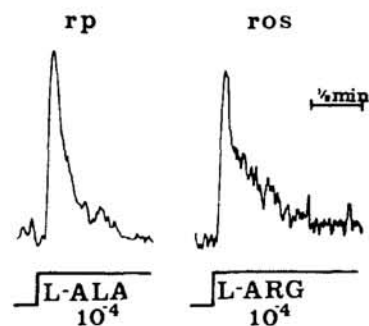

Fig. 2. Integrated, facial taste recordings to continuous application of amino acids to the palate and nasal barbel showing the phasic nature of the taste responses of the ramus palatinus (rp) and ramus ophthalmicus superficialis (ros), respectively.

Gustatory information from the extraoral and oral epithelium is "pipelined" into two separate subsystems, facial and glossopharyngeal-vagal, respectively. Each subsystem processes a subset of the incoming information (extraoral or oral) and coordinates a different component of food acquisition. Food search is accomplished by the extraoral subsystem, while selective ingestion is accomplished by the oral subsystem[2] (Fig. 3). The extraoral gustatory information terminates in the facial lobe where it is represented as a well-defined topographic map[9,10], while the oral information terminates in the adjacent vagal lobe where it is represented as a relatively diffuse map [11].

## II. ARCHITECTURE

The information represented in an information structure eventually requires an operating frame (architecture) within which to select and carry out the various processes. In ictalurid catfish, partially processed information from the primary gustatory centers (facial and vagal lobes) in the medullary region of the brainstem converges along ascending and descending pathways (Fig. 4). One of the centers in the ascending pathways is the secondary gustatory nucleus in the isthmic region which is connected to the corresponding nucleus of the opposite side via a large commissure[12,13]. Facial and vagal gustatory information crosses over to the opposite side via this commissure thus making it possible for neurons to extract information about interbarbel or interflank intensity differences. Although neurons in this region are known to have large receptive fields[14], the exact function of this large commissural nucleus is not yet clearly established.

It is quite clear, however, that gustatory information is at first "pipelined" into separate regions where it is processed in parallel[15] before converging onto neurons in the ascending (isthmic) and descending (reticular) processors as well as other regions within the medulla. The "pipelined" architecture underscores the need for differential processing of subsets of sensory inputs which are consequently integrated to coordinate temporal transitions between the various components of goal-directed behavior.

## III. SEARCH

An important task underlying all "intelligent" goal directed activity is that of search. In artificial systems this involves application of several general problem-solving methods such as means-end analysis, generate and test methods and heuristic search methods. No attempt, as yet, has been made to fit any of these models to the food-tracking behavior of the catfish. However, behavioral observations suggest that the catfish uses a combinatorial approach resulting in a different yet optimal foraging strategy each time [3].

What is interesting about biological models is that the intrinsic search strategy is expressed extrinsically by the behavior of the animal which, with a few precautions, can be observed quite easily. In addition, simple manipulations of either the animal[3] or its environment can provide interesting data about the search

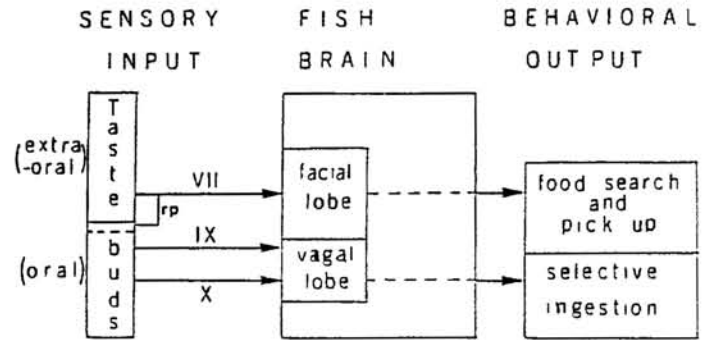

Fig. 3.

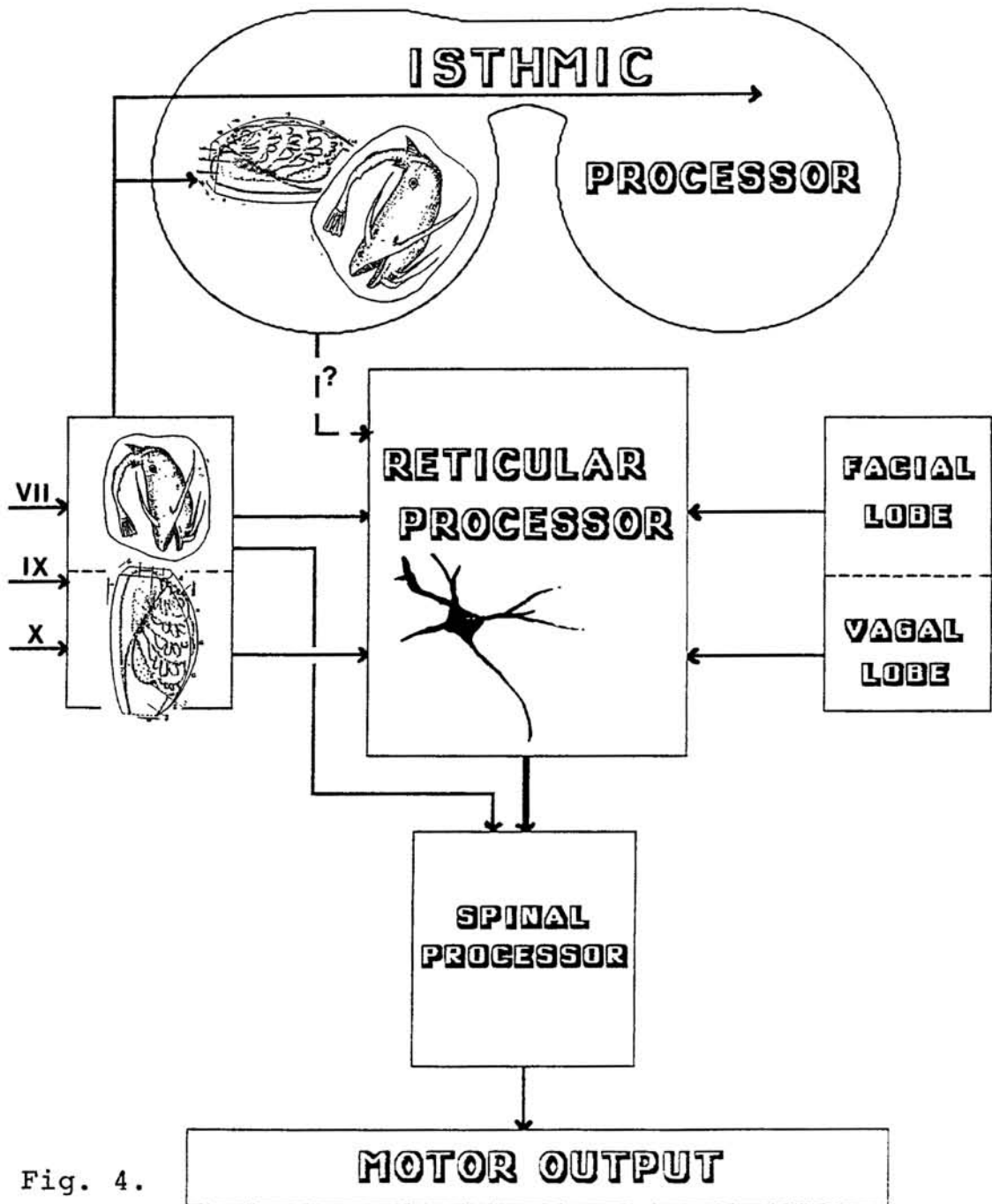

Fig. 4.

strategy/ies being used by the animal, which in turn can highlight some of the computational (neuronal) search strategies adopted by the brain e.g. the catfish seems to minimize the probability of failure by continuously interacting with the environment so as to be able to correct any computational or knowledge-based errors.

IV. KNOWLEDGE

If an "intelligent" goal-directed system resets to zero knowledge before each search trial, its success would depend entirely upon the information obtained over the time period of a search. Such a system would also require a labile architecture to process the varying sets of information generated during each search. For such a system, the solution space can become very large and given the constraints of time (generally an important criterion in biological systems) this can lead to continuous failure. For these reasons, knowledge becomes an important ingredient of an "intelligent" agent since it can keep the search under control.

For the gustatory system of the catfish too, randomly accessible knowledge, in combination with the immediately available information about the target, may play a critical role in the adoption of a successful search strategy. Although a significant portion of this knowledge is probably learned, it is not yet clear where and how this knowledge is stored in the catfish brain. The reduction in the solution space for a catfish which has gradually learned to find food in its environment may be attributed to the increase in the amount of knowledge, which to some extent may involve a restructuring of the neural networks during development.

## EXPERIMENTAL METHODS

The methods employed for the present study are only briefly introduced here. Neuroanatomical tracing techniques exploit the phenomenon of axonal transport. Crystals of the enzyme, horseradish peroxidase (HRP) or some other substance, when injected at a small locus in the brain, are taken up by the damaged neurons and transported anterogradely and retrogradely from cell bodies and/or axons at the injection site. In the present study, small superficial injections of HRP (Sigma, Type VI) were made at various loci in the facial lobe (FL) in separate animals. After a survival period of 3 to 5 days, the animals were sacrificed and the brains sectioned and reacted for visualization of the neuronal tracer. In this manner, complex neural circuits can be gradually delineated.

Electrophysiological recordings from neurons in the central nervous system were obtained using heat-pulled glass micropipettes. These glass electrodes had a tip diameter of approximately 1 μm and an impedance of less than 1 megohm when filled with an electrolyte (3M KCl or 3M Nacl).

Chemical stimulation of the receptive fields was accomplished by injection of stimuli (amino acids, amino acid mixtures and liver or bait-extract solutions) into a continuous flow of well-water over the receptive epithelium. Tactile stimulation was performed by gentle strokes of a sable hair brush or a glass probe.

EXPERIMENTAL OBSERVATIONS

Injections of HRP into the spinal cord labelled two relevant populations of cells, (i) in the ipsilateral reticular formation at the level of the facial lobe (FL), and (ii) a few large scattered cells within the ipsilateral, rostral portion of the lateral lobule of the FL (Fig. 5). Injection of HRP at several sites within the FL resulted in the identification of a small region in the FL from where anterogradely filled fibers project to the reticular formation (Fig. 5). Superimposition of these injection sites onto the anatomical map of the extraoral surface of the catfish indicated that this small region, within the facial lobe, corresponds to the snout region of the extraoral surface.

FACIO-RETICULAR PROJECTIONS          FACIO- & RETICULO -SPINAL PROJECTIONS

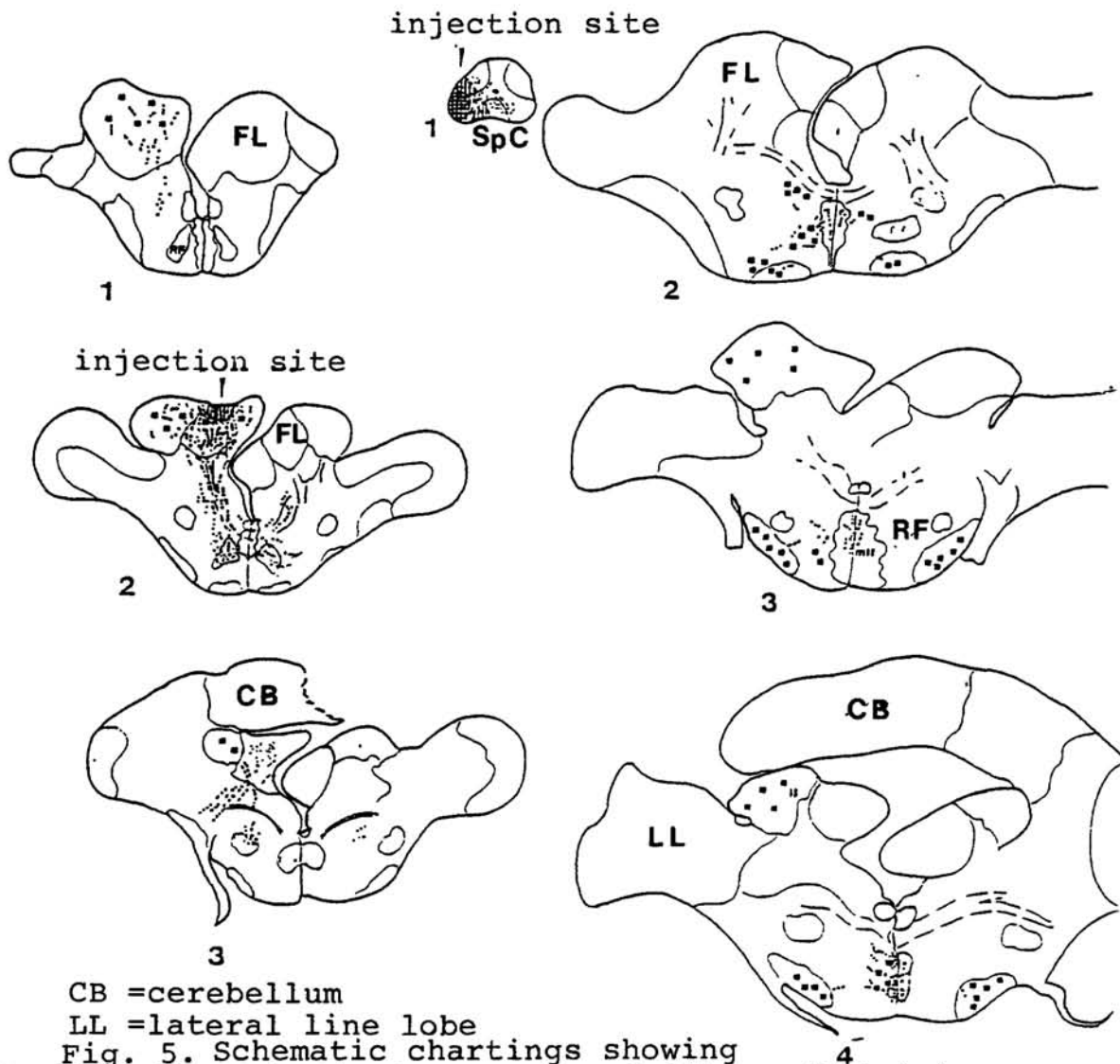

CB =cerebellum
LL =lateral line lobe
Fig. 5. Schematic chartings showing labelled cell bodies(squares) and fibers (dots) in transverse sections through the medulla.

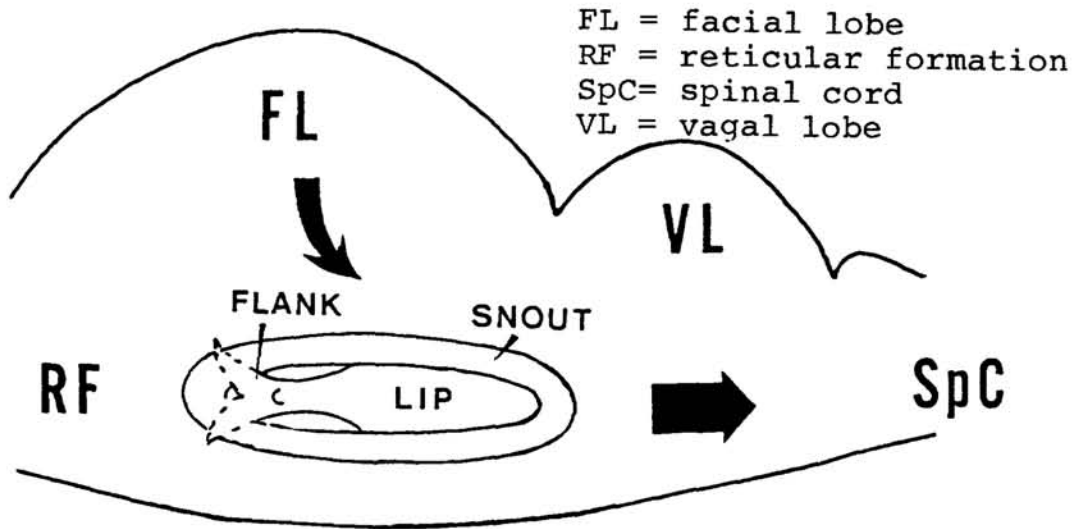

Fig. 6A.

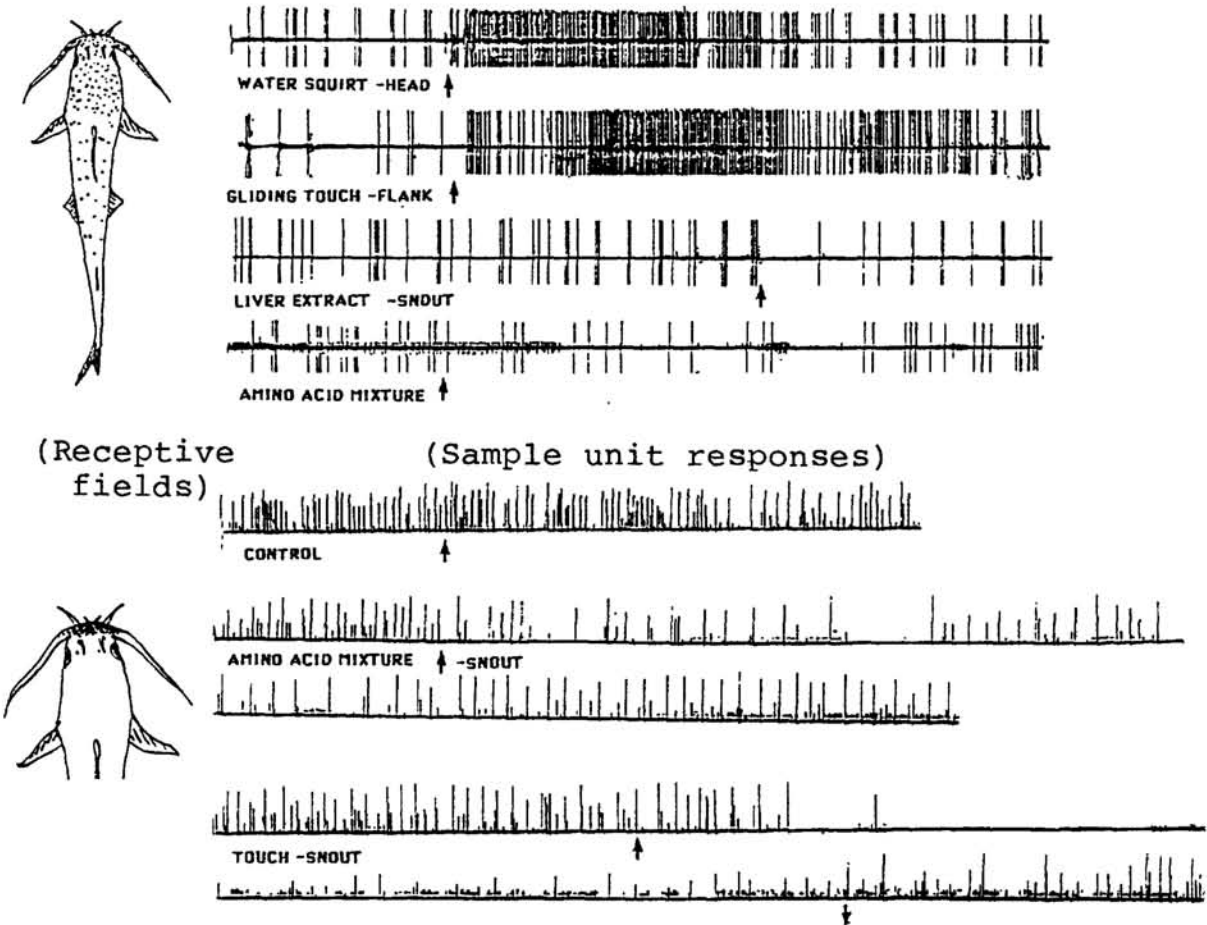

Fig. 6B.

Multiunit electrophysiological recordings from various anteroposterior levels of the reticular formation indicated that the snout region (upper lip and proximal portion of the maxillary barbels) of the catfish project to a disproportionately large region of the reticular formation along with a mixed representation of the flank (Fig. 6A).

Single unit recordings indicated that some neurons have receptive fields restricted to a bilateral portion of the snout region, while others had large receptive fields extending over the whole flank or over an anteroposterior half of the body (Fig. 6B).

## DISCUSSION

The experimental results obtained here suggest that facial lobe projections to the reticular formation form a functional connection. The reticular neurons project to the spinal cord and, most likely, influence the general cycle of swimming-related activity of motoneurons within the spinal cord [16].

The disproportionately large representation of the snout region within the medullary reticular formation, as determined electrophysiologically, is consistent with the anatomical data indicating that most of the fibers projecting to the reticular formation originate from cells in that portion of the facial lobe where the snout region is mapped. The lateral lobule of the spinal cord has a second pathway which projects directly into the spinal cord upto the level of the anterior end of the caudal fin and may coordinate reflexive turning.

The significance of the present results is best understood when considered together with previously known information about the anatomy and electrophysiology of the gustatory system. The information presented above is used to propose a model (Fig. 7) for a mechanism that may be involved during the homing phase of target tracking by the catfish. During homing, which refers to the last phase of target-tracking during food search, it may be assumed that the fish is rapidly approaching its target or moving through a steep signal intensity (stimulus concentration) gradient. The data presented above suggest that a neuronal mechanism exists which helps the catfish to lock on to the target during homing. This proposal is based upon the following considerations:

1. Owing to the rapidly adapting response of the peripheral filter, a tonic level of activity in the facial lobe input can occur only when the animal is moving through an increasing concentration gradient of the gustatory stimulus.

2. Facial lobe neurons, which receive inputs from the snout region, project to a group of cells in the reticular formation. Activity in the facio-reticular pathway causes a suppression in the spontaneous activity of the reticular neurons.

3. Direct and/or indirect spinal projections from the reticular neurons are involved in the modulation of activity of those spinal motoneurons which coordinate swimming. Thus, it may be hypothesized that during complete suppression of activity in a specific reticulo-spinal pathway, the fish swims straight ahead, but during excitation

of certain reticulospinal neurons the fish changes its direction as dictated by the pattern of activation.

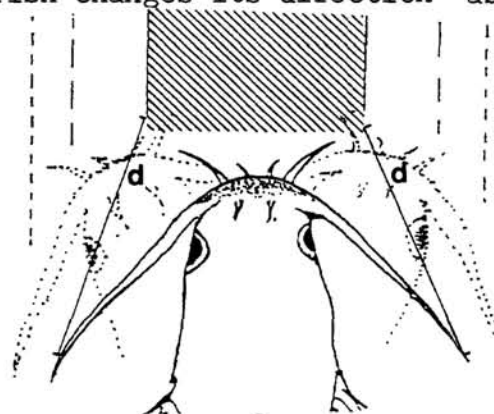

Fig. 7. The snout region of the catfish has special significance because of its extensive representation in the reticular formation. In case the fish makes a random or computational error, while approaching its target, the snout is the first region to move out of the stimulus gradient.

Thus, the spinal motoneurons, teleologically speaking, "seek" a gustatory stimulus in order to suppress activity of certain reticulospinal neurons, which in turn reduce variations in the pattern of activity of swimming-related spinal motoneurons. Accordingly, in a situation where the fish is rapidly approaching a target, ie. under the specific conditions of a continuously rising stimulus concentration at the snout region and an absence of a stimulus intensity difference across the barbels, there is a locking of the movement of the body (of the fish) towards the stationary or moving target (food or prey).

It should be pointed out, however, that the empirical data available so far, only offers clues to the target-tracking mechanism proposed here. Clearly, more research is needed to validate this proposal and to identify other mechanisms of target-tracking utilized by this biological system.

This research was supported in part by NIH Grant NS15258 to T.E. Finger.

## REFERENCES

1. P. B. Johnsen and J. H. Teeter, J. Comp. Physiol. 140, 95 (1981).
2. J. Atema, Brain Behav. and Evol. 4, 273-294, (1971).
3. J. E. Bardach, et al., Science, 155, 1276-1278, (1967).
4. A. Newell, Mc-Graw Hill Encyclopedia of Electronics and Computers, (1984), p.71-74.
5. C. J. Herrick, Bull. US. Fish. Comm. 22, 237-272, (1904).
6. J. Caprio, Comp. Biochem. Physiol. 52A, 247-251, (1975).
7. C. J. Davenport and J. Caprio, J. Comp. Physiol. 147, 217 (1982).
8. J. S. Kanwal and J. Caprio, Brain Res. 406, 105-112, (1987).
9. T. E. Finger, J. Comp. Neurol. 165, 513-526 (1976).
10. T. Marui and J. Caprio, Brain Res. 231, 185-190 (1982).
11. J. S. Kanwal and J. Caprio, J. Neurobiol. in press, (1988).
12. C. J. Herrick, J. Comp. Neurol. 15, 375-456 (1905).
13. C. J. Herrick, J. Comp. Neurol. 16, 403-440 (1906).
14. C. F. Lamb and J. Caprio, ISOT, #P70, (1986).
15. T. E. Finger and Y. Morita, Science, 227, 776-778 (1985).
16. P. S. G. Stein, Handbook of the Spinal Cord, (Marcel Dekker Inc., N.Y., 1984), p. 647.
